# Potential Boosters ?

**Nigel Duffy**
Department of Computer Science
University of California
Santa Cruz, CA 95064
*nigeduff@cse.ucsc.edu*

**David Helmbold**
Department of Computer Science
University of California
Santa Cruz, CA 95064
*dph@cse.ucsc.edu*

## Abstract

Recent interpretations of the Adaboost algorithm view it as performing a gradient descent on a potential function. Simply changing the potential function allows one to create new algorithms related to AdaBoost. However, these new algorithms are generally not known to have the formal boosting property. This paper examines the question of which potential functions lead to new algorithms that are boosters. The two main results are general sets of conditions on the potential; one set implies that the resulting algorithm is a booster, while the other implies that the algorithm is not. These conditions are applied to previously studied potential functions, such as those used by LogitBoost and Doom II.

## 1 Introduction

The first boosting algorithm appeared in Rob Schapire's thesis [1]. This algorithm was able to boost the performance of a weak PAC learner [2] so that the resulting algorithm satisfies the strong PAC learning [3] criteria. We will call any method that builds a strong PAC learning algorithm from a weak PAC learning algorithm a *PAC boosting algorithm*. Freund and Schapire later found an improved PAC boosting algorithm called AdaBoost [4], which also tends to improve the hypotheses generated by practical learning algorithms [5].

The AdaBoost algorithm takes a labeled training set and produces a *master hypothesis* by repeatedly calling a given learning method. The given learning method is used with different distributions on the training set to produce different *base hypotheses*. The master hypothesis returned by AdaBoost is a weighted vote of these base hypotheses. AdaBoost works iteratively, determining which examples are poorly classified by the current weighted vote and selecting a distribution on the training set to emphasize those examples.

Recently, several researchers [6, 7, 8, 9, 10] have noticed that Adaboost is performing a constrained gradient descent on an exponential potential function of the margins of the examples. The margin of an example is $yF(x)$ where $y$ is the $\pm 1$ valued label of the example $x$ and $F(x) \in \Re$ is the net weighted vote of master hypothesis $F$. Once Adaboost is seen this way it is clear that further algorithms may be derived by changing the potential function [6, 7, 9, 10].

The exponential potential used by AdaBoost has the property that the influence of a data point increases exponentially if it is repeatedly misclassified by the base hypotheses. This concentration on the "hard" examples allows AdaBoost to rapidly obtain a consistent hypothesis (assuming that the base hypotheses have certain properties). However, it also means that an incorrectly labeled or noisy example can quickly attract much of the distribution. It appears that this lack of noise-tolerance is one of AdaBoost's few drawbacks [11]. Several researchers [7, 8, 9, 10] have proposed potential functions which do not concentrate as much on these "hard" examples. However, they generally do not show that the derived algorithms have the PAC boosting property.

In this paper we return to the original motivation behind boosting algorithms and ask: "for which potential functions does gradient descent lead to PAC boosting algorithms" (i.e. boosters that create strong PAC learning algorithms from arbitrary weak PAC learners). We give necessary conditions that are met by some of the proposed potential functions (most notably the LogitBoost potential introduced by Friedman *et al.* [7]). Furthermore, we show that simple gradient descent on other proposed potential functions (such as the sigmoidal potential used by Mason *et al.* [10]) cannot convert arbitrary weak PAC learning algorithms into strong PAC learners. The aim of this work is to identify properties of potential functions required for PAC boosting, in order to guide the search for more effective potentials.

Some potential functions have an additional tunable parameter [10] or change over time [12]. Our results do not yet apply to such dynamic potentials.

## 2 PAC Boosting

Here we define the notions of PAC learning[1] and boosting, and define the notation used throughout the paper.

A *concept* $C$ is a subset of the learning domain $\mathcal{X}$. A *random example* of $C$ is a pair ($x \in \mathcal{X}, y \in \{-1, +1\}$) where $x$ is drawn from some distribution on $\mathcal{X}$ and $y = 1$ if $x \in C$ and $-1$ otherwise. A *concept class* is a set of concepts.

**Definition 1** *A (strong) PAC learner for concept class $C$ has the property that for every distribution $D$ on $\mathcal{X}$, all concepts $C \in \mathcal{C}$, and all $0 < \epsilon, \delta < 1/2$: with probability at least $1 - \delta$ the algorithm outputs a hypothesis $h$ where $\mathbf{P}_D[h(x) \neq C(x)] \leq \epsilon$. The learning algorithm is given $C$, $\epsilon$, $\delta$, and the ability to draw random examples of $C$ (w.r.t. distribution $D$), and must run in time bounded by* poly($1/\epsilon, 1/\delta$).

**Definition 2** *A weak PAC learner is similar to a strong PAC learner, except that it need only satisfy the conditions for a particular $0 < \epsilon_0, \delta_0 < 1/2$ pair, rather than for all $\epsilon, \delta$ pairs.*

**Definition 3** *A PAC boosting algorithm is a generic algorithm which can leverage any weak PAC learner to meet the strong PAC learning criteria.*

In the remainder of the paper we emphasize boosting the accuracy $\epsilon$ as it is much easier to boost the confidence $\delta$, see Haussler *et al.* [13] and Freund [14] for details. Furthermore, we emphasize boosting by re-sampling, where the strong PAC learner draws a large sample, and each iteration the weak learning algorithm is called with some distribution over this sample.

Throughout the paper we use the following notation.

- $m$ is the cardinality of the fixed sample $\{(x_1, y_1), \ldots, (x_m, y_m)\}$.
- $h_t(x)$ is the $\pm 1$ valued weak hypothesis created at iteration $t$.
- $\alpha_t$ is the weight or vote of $h_t$ in the master hypothesis, the $\alpha$'s may or may not be normalized so that $\sum_{t'=1}^{t} \alpha_{t'} = 1$.
- $F_t(x) = \sum_{t'=1}^{t}(\alpha_{t'} h_{t'}(x) / \sum_{\tau=1}^{t} \alpha_\tau) \in \Re$, is the master hypothesis[2] at iteration $t$.
- $u_{i,t} = y_i \sum_{t'=1}^{t} \alpha_{t'} h_{t'}(x)$ is the margin of $x_i$ after iteration $t$; the $t$ subscript is often omitted. Note that the margin is positive when the master hypothesis is correct, and the *normalized margin* is $u_{i,t} / \sum_{t'=1}^{t} \alpha_{t'}$.
- $p(u)$ is the *potential* of an instance with margin $u$, and the total potential is $\sum_{i=1}^{m} p(u_i)$.
- $\mathbf{P}_D[\ ], \mathbf{P}_S[\ ]$, and $\mathbf{E}_S[\ ]$ are the probability with respect to the unknown distribution over the domain, and the probability and expectations with respect to the uniform distribution over the sample, respectively.

Our results apply to total potential functions of the form $\sum_{i=1}^{m} p(u_i)$ where $p$ is positive and strictly decreasing.

## 3  Leveraging Learners by Gradient Descent

AdaBoost [4] has recently been interpreted as gradient descent independently by several groups [6, 7, 8, 9, 10]. Under this interpretation AdaBoost is seen as minimizing the total potential $\sum_{i=1}^{m} p(u_i) = \sum_{i=1}^{m} \exp(-u_i)$ via feasible direction gradient descent. On each iteration $t + 1$, AdaBoost chooses the direction of steepest descent as the distribution on the sample, and calls the weak learner to obtain a new base hypothesis $h_{t+1}$. The weight $\alpha_{t+1}$ of this new weak hypothesis is calculated to minimize[3] the resulting potential $\sum_{i=1}^{m} p(u_{i,t+1}) = \sum_{i=1}^{m} \exp(-(u_{i,t} + \alpha_{t+1} y_i h_{t+1}(x_i)))$.

This gradient descent idea has been generalized to other potential functions [6, 7, 10]. Duffy *et al.* [9] prove bounds for a similar gradient descent technique using a non-componentwise, non-monotonic potential function.

Note that if the weak learner returns a good hypothesis $h_t$ (with training error at most $\epsilon < 1/2$), then $\sum_{i=1}^{m} D_t(x_i) y_i h_t(x_i) > 1 - 2\epsilon > 0$. We set $r = 1 - 2\epsilon$, and assume that each base hypothesis produced satisfies $\sum_{i=1}^{m} D_t(x_i) y_i h_t(x_i) \geq r$.

In this paper we consider this general gradient descent approach applied to various potentials $\sum_{i=1}^{m} p(u_i)$. Note that each potential function $p$ has two corresponding gradient descent algorithms (see [6]). The *un-normalized* algorithms (like AdaBoost) continually add in new weak hypotheses while preserving the old $\alpha$'s. The *normalized* algorithms re-scale the $\alpha$'s so that they always sum to 1. In general, we call such algorithms "leveraging algorithms", reserving the term "boosting" for those that actually have the PAC boosting property.

## 4  Potentials that Don't Boost

In this section we describe sufficient conditions on potential functions so that the corresponding leveraging algorithm does not have the PAC boosting property. We

apply these conditions to show that two potentials from the literature do not lead to boosting algorithms.

**Theorem 1** *Let $p(u)$ be a potential function for which:*
 *1) the derivative, $p'(u)$, is increasing $(-p'(u)$ decreasing) in $\Re_+$, and*
 *2) $\exists \beta > 0$ such that for all $u > 0$, $-\beta p'(u) \geq -p'(-2u)$.*
*Then neither the normalized nor the un-normalized leveraging algorithms corresponding to potential $p$ have the PAC boosting property.*

This theorem is proven by an adversary argument. Whenever the concept class is sufficiently rich[4], the adversary can keep a constant fraction of the sample from being correctly labeled by the master hypothesis. Thus as the error tolerance $\epsilon$ goes to zero, the master hypotheses will not be sufficiently accurate.

We now apply this theorem to two potential functions from the literature.

Friedman *et al.* [7] describe a potential they call "Squared Error(p)" where the potential at $x_i$ is $\left( \dfrac{y_i + 1}{2} - \dfrac{e^{F(x_i)}}{e^{F(x_i)} + e^{-F(x_i)}} \right)^2$. This potential can be re-written as $p_{SE}(u_i) = \dfrac{1}{4} \left( 1 + 2 \dfrac{e^{-u_i} - e^{u_i}}{e^{u_i} + e^{-u_i}} + \left( \dfrac{e^{-u_i} - e^{u_i}}{e^{u_i} + e^{-u_i}} \right)^2 \right)$.

**Corollary 1** *Potential "Squared Error(p)" does not lead to a boosting algorithm.*

**Proof:** This potential satisfies the conditions of Theorem 1. It is strictly decreasing, and the second condition holds for $\beta = 2$.

Mason *et al.* [10] examine a normalized algorithm using the potential $p_D(u) = 1 - \tanh(\lambda u)$. Their algorithm optimizes over choices of $\lambda$ via cross-validation, and uses weak learners with slightly different properties. However, we can plug this potential directly into the gradient descent framework and examine the resulting algorithms.

**Corollary 2** *The DOOMII potential $p_D$ does not lead to a boosting algorithm for any fixed $\lambda$.*

**Proof:** The potential is strictly decreasing, and the second condition of Theorem 1 holds for $\beta = 1$.

Our techniques show that potentials that are sigmoidal in nature do not lead to algorithms with the PAC boosting property. Since sigmoidal potentials are generally better over-estimates of the $0, 1$ loss than the potential used by AdaBoost, our results imply that boosting algorithms must use a potential with more subtle properties than simply upper bounding the $0, 1$ loss.

## 5 Potential Functions That Boost

In this section we give sufficient conditions on a potential function for it's corresponding un-normalized algorithm to have the PAC boosting property. This result implies that AdaBoost [4] and LogitBoost [7] have the PAC boosting property (Although this was previously known for AdaBoost [4], we believe this is a new result for LogitBoost).

One set of conditions on the potential imply that it decreases roughly exponentially when the (un-normalized) margins are large. Once the margins are in this exponential region, ideas similar to those used in AdaBoost's analysis show that the minimum *normalized* margin quickly becomes bounded away from zero. This allows us to bound the generalization error using a theorem from Bartlett *et al.* [15].

A second set of conditions governs the behavior of the potential function before the un-normalized margins are large enough. These conditions imply that the total potential decreases by a constant factor each iteration. Therefore, too much time will not be spent before all the margins enter the exponential region.

The margin value bounding the exponential region is $U$, and once $\sum_{i=1}^{t} p(u_i) \leq p(U)$, all margins $p(u_i)$ will remain in the exponential region. The following theorem gives conditions on $p$ ensuring that $\sum_{i=1}^{t} p(u_i)$ quickly becomes less than $p(U)$.

**Theorem 2** *If the following conditions hold for $p(u)$ and $U$:*

1. *$-p'(u)$ is strictly decreasing and $0 < p''(u) \leq B$, and*

2. *$\exists q > 0$ such that $p(u) \leq -qp'(u) \; \forall u > U$,*

*then $\sum_{i=1}^{m} p(u_i) \leq p(U)$ after $T_1 \geq \dfrac{4Bq^2m^2p(0)\ln\left(\frac{mp(0)}{p(U)}\right)}{p(U)^2r^2}$ iterations.*

The proof of this theorem approximates the new total potential by the old potential minus $\alpha$ times a linear term, plus an error. By bounding the error as a function of $\alpha$ and minimizing we demonstrate that some values of $\alpha$ give a sufficient decrease in the total potential.

**Theorem 3** *If the following conditions hold for $p(u)$, $U$, $q$, and iteration $T_1$:*

1. *$\exists \beta \geq \sqrt{3}$ such that $-p'(u+v) \leq p(u+v) \leq -p'(u)\beta^{-v}q$ whenever $-1 \leq v \leq 1$ and $u > U$,*

2. *$\sum_{i=1}^{m} p(u_{i,T_1}) \leq p(U)$,*

3. *$-p'(u)$ is strictly decreasing, and*

4. *$\exists C > 0, \gamma > 1$ such that $Cp(u) \geq \gamma^{-u} \; \forall u > U$*

*then $\mathbf{P}_S[yF_T(x) \leq \theta] \leq \dfrac{qC}{m}\gamma^{\theta T_1}p(U)\left(\gamma^{\theta}q\sqrt{1-r^2}\right)^{(T-T_1)}$ which decreases exponentially in $T - T_1$ if $\theta < \dfrac{-\ln(q\sqrt{1-r^2})}{\ln(\gamma)}$.*

The proof of this theorem is a generalization of the AdaBoost proof.

Combining these two theorems, and the generalization bound from Theorem 2 of Bartlett *et al.* [15] gives the following result, where $d$ is the VC dimension of the weak hypothesis class.

**Theorem 4** *If for all edges $0 < r < 1/2$ there exists $T_{1,r} \leq \text{poly}(m, 1/r)$, $U_r$, and $q_r$ satisfying the conditions of Theorem 3 such that $p(U_r) \geq \text{poly}(r)$ and $q_r\sqrt{1-r^2} = l(r) < 1 - \text{poly}(r)$, then in time $\text{poly}(m, 1/r)$ all examples have nor-*

*malized margin at least* $\theta = \ln\left(\frac{(l(r)+1)}{2l(r)}\right) / \ln(\gamma)$ *and*

$$\mathbf{P}_D[yF_T(x) \le 0] \in O\left(\frac{1}{\sqrt{m}}\left(\frac{\ln^2(\gamma)d\log^2(m/d)}{(\ln(l(r)+1) - \ln(2l(r)))^2} + \log(1/\delta)\right)^{\frac{1}{2}}\right) .$$

*Choosing m appropriately makes the error rate sufficiently small so that the algorithm corresponding to p has the PAC boosting property.*

We now apply Theorem 4 to show that the AdaBoost and LogitBoost potentials lead to boosting algorithms.

## 6  Some Boosting Potentials

In this section we show as a direct consequence of our Theorem 4 that the potential functions for AdaBoost and LogitBoost lead to boosting algorithms. Note that the LogitBoost algorithm we analyze is not exactly the same as that described by Friedman *et al.* [7], their "weak learner" optimizes a square loss which appears to better fit the potential. First we re-derive the boosting property for AdaBoost.

**Corollary 3** *AdaBoost's [16] potential boosts.*

**Proof:** To prove this we simply need to show that the potential $p(u) = \exp(-u)$ satisfies the conditions of Theorem 4. This is done by setting $U_r = -\ln(m)$, $q_r = 1$, $\gamma = \beta = e$, $C = 1$, and $T_1 = 0$.

**Corollary 4** *The log-likelihood potential (as used in LogitBoost [7]) boosts.*

**Proof:** In this case $p(u) = \ln(1 + e^{-u})$ and $-p'(u) = \frac{e^{-u}}{1+e^{-u}}$. We set $\gamma = \beta = e$, $C = 2$, $U_r = -\ln\left(\frac{\sqrt{1 - \epsilon^2/2}}{\sqrt{1-\epsilon^2}} - 1\right)$ and $q_r = 1 + \exp(-U_r) = \frac{\sqrt{1-\epsilon^2/2}}{\sqrt{1-\epsilon^2}}$. Now Theorem 2 shows that after $T_1 \le \text{poly}(m, 1/r)$ iterations the conditions of Theorem 4 are satisfied.

## 7  Conclusions

In this paper we have examined leveraging weak learners using a gradient descent approach [9]. This approach is a direct generalization of the Adaboost [4, 16] algorithm, where Adaboost's exponential potential function is replaced by alternative potentials. We demonstrated properties of potentials that are sufficient to show that the resulting algorithms are PAC boosters, and other properties that imply that the resulting algorithms are not PAC boosters. We applied these results to several potential functions from the literature [7, 10, 16].

New insight can be gained from examining our criteria carefully. The conditions that show boosting leave tremendous freedom in the choice of potential function for values less than some $U$, perhaps this freedom can be used to choose potential functions which do not overly concentrate on noisy examples.

There is still a significant gap between these two sets of properties, we are still a long way from classifying arbitrary potential functions as to their boosting properties.

There are other classes of leveraging algorithms. One class looks at the distances between successive distributions [17, 18]. Another class changes their potential

over time [6, 8, 12, 14]. The criteria for boosting may change significantly with these different approaches. For example, Freund recently presented a boosting algorithm [12] that uses a time-varying sigmoidal potential. It would be interesting to adapt our techniques to such dynamic potentials.

## Footnotes

[1]To simplify the presentation we omit the instance space dimension and target representation length parameters.

[2]The prediction of the master hypothesis on instance $x$ is the sign of $F_t(x)$.

[3]Our current proofs require that the actual $\alpha_t$'s be no greater than a constant (say 1). Therefore, this minimizing $\alpha$ may need to be reduced.

[4]The VC-dimension 4 concept class consisting of pairs of intervals on the real line is sufficient for our adversary.

# References

[1] Robert E. Schapire. *The Design and Analysis of Efficient Learning Algorithms*. MIT Press, 1992.

[2] Michael Kearns and Leslie Valiant. Cryptographic limitations on learning Boolean formulae and finite automata. *Journal of the ACM*, 41(1):67–95, January 1994.

[3] L. G. Valiant. A theory of the learnable. *Commun. ACM*, 27(11):1134–1142, November 1984.

[4] Yoav Freund and Robert E. Schapire. A decision-theoretic generalization of on-line learning and an application to boosting. *Journal of Computer and System Sciences*, 55(1):119–139, August 1997.

[5] Eric Bauer and Ron Kohavi. An empirical comparison of voting classification algorithms: Bagging, boosting and variants. *Machine Learning*, 36(1–2):105–39, 1999.

[6] Leo Breiman. Arcing the edge. Technical Report 486, Department of Statistics, University of California, Berkeley., 1997. available at www.stat.berkeley.edu.

[7] Jerome Friedman, Trevor Hastie, and Robert Tibshirani. Additive logistic regression: a statistical view of boosting. Technical report, Stanford University, 1998.

[8] G. Rätsch, T. Onoda, and K.-R. Müller. Soft margins for AdaBoost. *Machine Learning*, 2000. To appear.

[9] Nigel Duffy and David P. Helmbold. A geometric approach to leveraging weak learners. In Paul Fischer and Hans Ulrich Simon, editors, *Computational Learning Theory: 4th European Conference (EuroCOLT '99)*, pages 18–33. Springer-Verlag, March 1999.

[10] Llew Mason, Jonathan Baxter, Peter Bartlett, and Marcus Frean. Boosting algorithms as gradient descent. To appear in NIPS 2000.

[11] Thomas G. Dietterich. An experimental comparison of three methods for constructing ensembles of decision trees: Bagging, Boosting, and Randomization. *Machine Learning*. To appear.

[12] Yoav Freund. An adaptive version of the boost-by-majority algorithm. In *Proc. 12th Annu. Conf. on Comput. Learning Theory*, pages 102–113. ACM, 1999.

[13] David Haussler, Michael Kearns, Nick Littlestone, and Manfred K. Warmuth. Equivalence of models for polynomial learnability. *Information and Computation*, 95(2):129–161, December 1991.

[14] Y. Freund. Boosting a weak learning algorithm by majority. *Information and Computation*, 121(2):256–285, September 1995.

[15] Robert E. Schapire, Yoav Freund, Peter Bartlett, and Wee Sun Lee. Boosting the margin: A new explanation for the effectiveness of voting methods. *The Annals of Statistics*, 26(5):1651–1686, 1998.

[16] Robert E. Schapire and Yoram Singer. Improved boosting algorithms using confidence-rated predictions. *Machine Learning*, 37(3):297–336, December 1999.

[17] Jyrki Kivinen and Manfred K. Warmuth. Boosting as entropy projection. In *Proc. 12th Annu. Conf. on Comput. Learning Theory*, pages 134–144. ACM, 1999.

[18] John Lafferty. Additive models, boosting, and inference for generalized divergences. In *Proc. 12th Annu. Conf. on Comput. Learning Theory*, pages 125–133. ACM.
